# Differential Entropic Clustering of Multivariate Gaussians

**Jason V. Davis**      **Inderjit Dhillon**
Dept. of Computer Science
University of Texas at Austin
Austin, TX 78712
{jdavis,inderjit}@cs.utexas.edu

## Abstract

Gaussian data is pervasive and many learning algorithms (e.g., $k$-means) model their inputs as a *single* sample drawn from a multivariate Gaussian. However, in many real-life settings, each input object is best described by *multiple* samples drawn from a multivariate Gaussian. Such data can arise, for example, in a movie review database where each movie is rated by several users, or in time-series domains such as sensor networks. Here, each input can be naturally described by both a mean vector *and* covariance matrix which parameterize the Gaussian distribution. In this paper, we consider the problem of clustering such input objects, each represented as a multivariate Gaussian. We formulate the problem using an information theoretic approach and draw several interesting theoretical connections to Bregman divergences and also Bregman *matrix* divergences. We evaluate our method across several domains, including synthetic data, sensor network data, and a statistical debugging application.

## 1   Introduction

Gaussian data is pervasive in all walks of life and many learning algorithms—e.g. $k$-means, principal components analysis, linear discriminant analysis, etc—model each input object as a *single* sample drawn from a multivariate Gaussian. For example, the $k$-means algorithm assumes that each input is a single sample drawn from one of $k$ (unknown) isotropic Gaussians. The goal of $k$-means can be viewed as the discovery of the mean of each Gaussian and recovery of the generating distribution of each input object.

However, in many real-life settings, each input object is naturally represented by *multiple* samples drawn from an underlying distribution. For example, a student's scores in reading, writing, and arithmetic can be measured at each of four quarters throughout the school year. Alternately, consider a website where movies are rated on the basis of originality, plot, and acting. Here, several different users may rate the same movie. Multiple samples are also ubiquitous in time-series data such as sensor networks, where each sensor device continually monitors its environmental conditions (e.g. humidity, temperature, or light). Clustering is an important data analysis task used in many of these applications. For example, clustering sensor network devices has been used for optimizing routing of the network and also for discovering trends between sensor nodes. If the $k$-means algorithm is employed, then only the means of the distributions will be clustered, ignoring all second order covariance information. Clearly, a better solution is needed.

In this paper, we consider the problem of clustering input objects, each of which can be represented by a multivariate Gaussian distribution. The "distance" between two Gaussians can be quantified in an information theoretic manner, in particular by their differential relative entropy. Interestingly, the differential relative entropy between two multivariate Gaussians can be expressed as the convex combination of two Bregman divergences—a Mahalanobis distance between mean vectors and

a Burg matrix divergence between the covariance matrices. We develop an EM style clustering algorithm and show that the optimal cluster parameters can be cheaply determined via a simple, closed-form solution. Our algorithm is a Bregman-like clustering method that clusters both means and covariances of the distributions in a unified framework.

We evaluate our method across several domains. First, we present results from synthetic data experiments, and show that incorporating second order information can dramatically increase clustering accuracy. Next, we apply our algorithm to a real-world sensor network dataset comprised of 52 sensor devices that measure temperature, humidity, light, and voltage. Finally, we use our algorithm as a statistical debugging tool by clustering the behavior of functions in a program across a set of known software bugs.

## 2 Preliminaries

We first present some essential background material. The *multivariate Gaussian* distribution is the multivariate generalization of the standard univariate case. The probability density function (pdf) of a $d$-dimensional multivariate Gaussian is parameterized by mean vector $\boldsymbol{\mu}$ and positive definite covariance matrix $\boldsymbol{\Sigma}$:

$$p(\boldsymbol{x}|\boldsymbol{\mu}, \boldsymbol{\Sigma}) = \frac{1}{(2\pi)^{\frac{d}{2}}|\boldsymbol{\Sigma}|^{\frac{1}{2}}} \exp\left(-\frac{1}{2}(\boldsymbol{x} - \boldsymbol{\mu})^T \boldsymbol{\Sigma}^{-1}(\boldsymbol{x} - \boldsymbol{\mu})\right),$$

where $|\boldsymbol{\Sigma}|$ is the determinant of $\boldsymbol{\Sigma}$.

The *Bregman divergence* [2] with respect to $\phi$ is defined as:

$$D_\phi(\boldsymbol{x}, \boldsymbol{y}) = \phi(\boldsymbol{x}) - \phi(\boldsymbol{y}) - (\boldsymbol{x} - \boldsymbol{y})^T \nabla\phi(\boldsymbol{y}),$$

where $\phi$ is a real-valued, strictly convex function defined over a convex set $Q = dom(\phi) \subset \mathbb{R}^d$ such that $\phi$ is differentiable on the relative interior of $Q$. For example, if $\phi(\boldsymbol{x}) = \boldsymbol{x}^T\boldsymbol{x}$, then the resulting Bregman divergence is the standard squared Euclidean distance. Similarly, if $\phi(\boldsymbol{x}) = \boldsymbol{x}^T\boldsymbol{A}^T\boldsymbol{A}\boldsymbol{x}$, for some arbitrary non-singular matrix $\boldsymbol{A}$, then the resulting divergence is the Mahalanobis distance $M_{\boldsymbol{S}^{-1}}(\boldsymbol{x}, \boldsymbol{y}) = (\boldsymbol{x} - \boldsymbol{y})^T\boldsymbol{S}^{-1}(\boldsymbol{x} - \boldsymbol{y})$, parameterized by the covariance matrix $\boldsymbol{S}$, $\boldsymbol{S}^{-1} = \boldsymbol{A}^T\boldsymbol{A}$. Alternately, if $\phi(\boldsymbol{x}) = \sum_i(x_i \log x_i - x_i)$, then the resulting divergence is the (unnormalized) relative entropy. Bregman divergences generalize many properties of squared loss and relative entropy.

Bregman divergences can be naturally extended to matrices, as follows:

$$D_\phi(\boldsymbol{X}, \boldsymbol{Y}) = \phi(\boldsymbol{X}) - \phi(\boldsymbol{Y}) - tr((\nabla\phi(\boldsymbol{Y}))^T(\boldsymbol{X} - \boldsymbol{Y})),$$

where $\boldsymbol{X}$ and $\boldsymbol{Y}$ are matrices, $\phi$ is a real-valued, strictly convex function defined over matrices, and $tr(\boldsymbol{A})$ denotes the trace of $\boldsymbol{A}$. Consider the function $\phi(\boldsymbol{X}) = \|\boldsymbol{X}\|_F^2$. Then the corresponding Bregman matrix divergence is the squared Frobenius norm, $\|\boldsymbol{X} - \boldsymbol{Y}\|_F^2$. The Burg matrix divergence is generated from a function of the *eigenvalues* $\lambda_1, ..., \lambda_d$ of the positive definite matrix $\boldsymbol{X}$: $\phi(\boldsymbol{X}) = -\sum_i \log \lambda_i = -\log|\boldsymbol{X}|$, the Burg entropy of the eigenvalues. The resulting Burg matrix divergence is:

$$B(\boldsymbol{X}, \boldsymbol{Y}) = tr(\boldsymbol{X}\boldsymbol{Y}^{-1}) - \log|\boldsymbol{X}\boldsymbol{Y}^{-1}| - d. \tag{1}$$

As we shall see later, the Burg matrix divergence will arise naturally in our application. Let $\lambda_1, ..., \lambda_d$ be the eigenvalues of $\boldsymbol{X}$ and $\boldsymbol{v_1}, ..., \boldsymbol{v_d}$ the corresponding eigenvectors and let $\gamma_1, ..., \gamma_d$ be the eigenvalues of $\boldsymbol{Y}$ with eigenvectors $\boldsymbol{w_1}, ..., \boldsymbol{w_d}$. The Burg matrix divergence can also be written as

$$B(\boldsymbol{X}, \boldsymbol{Y}) = \sum_i \sum_j \frac{\lambda_i}{\gamma_j}(\boldsymbol{v_i}^T\boldsymbol{w_j})^2 - \sum_i \log \frac{\lambda_i}{\gamma_i} - d.$$

From the first term above, we see that the Burg matrix divergence is a function of the eigenvalues as well as of the *eigenvectors* of $\boldsymbol{X}$ and $\boldsymbol{Y}$.

The *differential entropy* of a continuous random variable $\boldsymbol{x}$ with probability density function $f$ is defined as

$$h(f) = -\int f(\boldsymbol{x}) \log f(\boldsymbol{x}) d\boldsymbol{x}.$$

It can be shown [3] that an $n$-bit quantization of a continuous random variable with pdf $f$ has Shannon entropy approximately equal to $h(f) + n$. The continuous analog of the discrete relative

entropy is the differential relative entropy. Given a random variable $\boldsymbol{x}$ with pdf's $f$ and $g$, the differential relative entropy is defined as

$$D(f||g) = \int f(\boldsymbol{x}) \log \frac{f(\boldsymbol{x})}{g(\boldsymbol{x})} d\boldsymbol{x}.$$

## 3 Clustering Multivariate Gaussians via Differential Relative Entropy

Given a set of $n$ multivariate Gaussians parameterized by mean vectors $\boldsymbol{m_1}, ..., \boldsymbol{m_n}$ and covariances $\boldsymbol{S_1}, ..., \boldsymbol{S_n}$, we seek a disjoint and exhaustive partitioning of these Gaussians into $k$ different clusters, $\pi_1, ..., \pi_k$. Each cluster $j$ can be represented by a multivariate Gaussian parameterized by mean $\boldsymbol{\mu_j}$ and covariance $\boldsymbol{\Sigma_j}$. Using differential relative entropy as the distance measure between Gaussians, the problem of clustering may be posed as the minimization (over all clusterings) of

$$\sum_{j=1}^{k} \sum_{\{i:\pi_i=j\}} D(p(\boldsymbol{x}|\boldsymbol{m_i}, \boldsymbol{S_i})||p(\boldsymbol{x}|\boldsymbol{\mu_j}, \boldsymbol{\Sigma_j})). \tag{2}$$

### 3.1 Differential Relative Entropy and Multivariate Gaussians

We first show that the differential entropy between two multivariate Gaussians can be expressed as a convex combination of a Mahalanobis distance between means and the Burg matrix divergence between covariance matrices.

Consider two multivariate Gaussians, parameterized by mean vectors $\boldsymbol{m}$ and $\boldsymbol{\mu}$, and covariances $\boldsymbol{S}$ and $\boldsymbol{\Sigma}$, respectively. We first note that the differential relative entropy can be expressed as $D(f||g) = \int f \log f - f \log g = -h(f) - \int f \log g$. The first term is just the negative differential entropy of $p(\boldsymbol{x}|\boldsymbol{m}, \boldsymbol{S})$, which can be shown [3] to be:

$$h(p(\boldsymbol{x}|\boldsymbol{m}, \boldsymbol{S})) = \frac{d}{2} + \frac{1}{2} \log(2\pi)^d |\boldsymbol{S}|. \tag{3}$$

We now consider the second term:

$$
\begin{aligned}
\int p(\boldsymbol{x}|\boldsymbol{m}, \boldsymbol{S}) \log p(\boldsymbol{x}|\boldsymbol{\mu}, \boldsymbol{\Sigma}) &= \int p(\boldsymbol{x}|\boldsymbol{m}, \boldsymbol{S}) \left[ -\frac{1}{2}(\boldsymbol{x} - \boldsymbol{\mu})^T \boldsymbol{\Sigma}^{-1}(\boldsymbol{x} - \boldsymbol{\mu}) - \log(2\pi)^{\frac{d}{2}} |\boldsymbol{\Sigma}|^{\frac{1}{2}} \right] \\
&= -\frac{1}{2} \int p(\boldsymbol{x}|\boldsymbol{m}, \boldsymbol{S}) tr(\boldsymbol{\Sigma}^{-1}(\boldsymbol{x} - \boldsymbol{\mu})(\boldsymbol{x} - \boldsymbol{\mu})^T) \\
&\quad - \int p(\boldsymbol{x}|\boldsymbol{m}, \boldsymbol{S}) \log(2\pi)^{\frac{d}{2}} |\boldsymbol{\Sigma}|^{\frac{1}{2}} \\
&= -\frac{1}{2} tr \left( \boldsymbol{\Sigma}^{-1} \mathrm{E} \left[ (\boldsymbol{x} - \boldsymbol{\mu})(\boldsymbol{x} - \boldsymbol{\mu})^T \right] \right) - \frac{1}{2} \log(2\pi)^d |\boldsymbol{\Sigma}| \\
&= -\frac{1}{2} tr \left( \boldsymbol{\Sigma}^{-1} \mathrm{E} \left[ ((\boldsymbol{x} - \boldsymbol{m}) + (\boldsymbol{m} - \boldsymbol{\mu}))((\boldsymbol{x} - \boldsymbol{m}) + (\boldsymbol{m} - \boldsymbol{\mu}))^T \right] \right) \\
&\quad - \frac{1}{2} \log(2\pi)^d |\boldsymbol{\Sigma}| \\
&= -\frac{1}{2} tr \left( \boldsymbol{\Sigma}^{-1} \boldsymbol{S} + \boldsymbol{\Sigma}^{-1}(\boldsymbol{m} - \boldsymbol{\mu})(\boldsymbol{m} - \boldsymbol{\mu})^T \right) - \frac{1}{2} \log(2\pi)^d |\boldsymbol{\Sigma}| \\
&= -\frac{1}{2} tr \left( \boldsymbol{\Sigma}^{-1} \boldsymbol{S} \right) - \frac{1}{2}(\boldsymbol{m} - \boldsymbol{\mu})^T \boldsymbol{\Sigma}^{-1}(\boldsymbol{m} - \boldsymbol{\mu}) - \frac{1}{2} \log(2\pi)^d |\boldsymbol{\Sigma}|.
\end{aligned}
$$

The expectation above is taken over the distribution $p(\boldsymbol{x}|\boldsymbol{m}, \boldsymbol{S})$. The second to last line above follows from the definition of $\boldsymbol{\Sigma} = E[(\boldsymbol{x} - \boldsymbol{m})(\boldsymbol{x} - \boldsymbol{m})^T]$ and also from the fact that $E[(\boldsymbol{x} -$

$\boldsymbol{m})(\boldsymbol{m} - \boldsymbol{\mu})^T] = E[\boldsymbol{x} - \boldsymbol{m}](\boldsymbol{m} - \boldsymbol{\mu})^T = \boldsymbol{0}$. Thus, we have

$$
\begin{aligned}
D(p(\boldsymbol{x}|\boldsymbol{m}, \boldsymbol{S})||p(\boldsymbol{x}|\boldsymbol{\mu}, \boldsymbol{\Sigma})) &= -\frac{d}{2} - \frac{1}{2}\log(2\pi)^d|\boldsymbol{S}| + \frac{1}{2}tr(\boldsymbol{\Sigma}^{-1}\boldsymbol{S}) + \frac{1}{2}\log(2\pi)^d|\boldsymbol{\Sigma}| \qquad (4) \\
&\quad + \frac{1}{2}(\boldsymbol{m} - \boldsymbol{\mu})^T\boldsymbol{\Sigma}^{-1}(\boldsymbol{m} - \boldsymbol{\mu}) \\
&= \frac{1}{2}\left(tr(\boldsymbol{S}\boldsymbol{\Sigma}^{-1}) - \log|\boldsymbol{S}\boldsymbol{\Sigma}^{-1}| - d\right) + \frac{1}{2}(\boldsymbol{m} - \boldsymbol{\mu})^T\boldsymbol{\Sigma}^{-1}(\boldsymbol{m} - \boldsymbol{\mu}) \\
&= \frac{1}{2}B(\boldsymbol{S}, \boldsymbol{\Sigma}) + \frac{1}{2}M_{\boldsymbol{\Sigma}^{-1}}(\boldsymbol{m}, \boldsymbol{\mu}), \qquad (5)
\end{aligned}
$$

where $B(\boldsymbol{S}, \boldsymbol{\Sigma})$ is the Burg matrix divergence and $M_{\boldsymbol{\Sigma}^{-1}}(\boldsymbol{m}, \boldsymbol{\mu})$ is the Mahalanobis distance, parameterized by the covariance matrix $\boldsymbol{\Sigma}$.

We now consider the problem of finding the optimal representative Gaussian for a set of $c$ Gaussians with means $\boldsymbol{m_1}, ..., \boldsymbol{m_c}$ and covariances $\boldsymbol{S_1}, ..., \boldsymbol{S_c}$. For non-negative weights $\alpha_1, ...\alpha_c$ such that $\sum_i \alpha_i = 1$, the optimal representative minimizes the cumulative differential relative entropy:

$$
\begin{aligned}
p(\boldsymbol{x}|\boldsymbol{\mu}^*, \boldsymbol{\Sigma}^*) &= \arg\min_{p(\boldsymbol{x}|\boldsymbol{\mu}, \boldsymbol{\Sigma})} \sum_i \alpha_i D(p(\boldsymbol{x}|\boldsymbol{m_i}, \boldsymbol{S_i})||p(\boldsymbol{x}|\boldsymbol{\mu}, \boldsymbol{\Sigma})) \qquad (6) \\
&= \arg\min_{p(\boldsymbol{x}|\boldsymbol{\mu}, \boldsymbol{\Sigma})} \sum_i \alpha_i \left(\frac{1}{2}B(\boldsymbol{S_i}, \boldsymbol{\Sigma}) + \frac{1}{2}M_{\boldsymbol{\Sigma}^{-1}}(\boldsymbol{m_i}, \boldsymbol{\mu})\right). \qquad (7)
\end{aligned}
$$

The second term can be viewed as minimizing the Bregman information with respect to some fixed (albeit unknown) Bregman divergence (i.e. the Mahalanobis distance parameterized by some covariance matrix $\boldsymbol{\Sigma}$). Consequently, it has a unique minimizer [1] of the form

$$
\boldsymbol{\mu}^* = \sum_i \alpha_i \boldsymbol{m_i}. \qquad (8)
$$

Next, we note that equation (7) is strictly convex in $\boldsymbol{\Sigma}^{-1}$. Thus, we can derive the optimal $\boldsymbol{\Sigma}^*$ by setting the gradient of (7) with respect to $\boldsymbol{\Sigma}^{-1}$ to 0:

$$
\frac{\partial}{\partial\boldsymbol{\Sigma}^{-1}} \sum_{i=1}^n \alpha_i D(p(\boldsymbol{x}|\boldsymbol{m_i}, \boldsymbol{S_i})||p(\boldsymbol{x}|\boldsymbol{\mu}, \boldsymbol{\Sigma})) = \sum_{i=1}^n \alpha_i \left(\boldsymbol{S_i} - \boldsymbol{\Sigma} + (\boldsymbol{m_i} - \boldsymbol{\mu}^*)(\boldsymbol{m_i} - \boldsymbol{\mu}^*)^T\right).
$$

Setting this to zero yields

$$
\boldsymbol{\Sigma}^* = \sum_i \alpha_i \left(\boldsymbol{S_i} + (\boldsymbol{m_i} - \boldsymbol{\mu}^*)(\boldsymbol{m_i} - \boldsymbol{\mu}^*)^T\right). \qquad (9)
$$

Figure 1 illustrates optimal representatives of two 2-dimensional Gaussians with means marked by points A and B, and covariances outlined with solid lines. The optimal Gaussian representatives are denoted with dotted covariances; the representative on the left uses weights, $(\alpha_A = \frac{2}{3}, \alpha_B = \frac{1}{3})$, while the representative on the right uses weights $(\alpha_A = \frac{1}{3}, \alpha_B = \frac{2}{3})$. As we can see from equation (8), the optimal representative mean is the weighted average of the means of the constituent Gaussians. Interestingly, the optimal covariance turns out to be the average of the constituent covariances plus rank one updates. These rank-one changes account for the deviations from the individual means to the representative mean.

## 3.2 Algorithm

Algorithm 1 presents our clustering algorithm for the case where each Gaussian has equal weight $\alpha_i = \frac{1}{n}$. The method works in an EM-style framework. Initially, cluster assignments are chosen (these can be assigned randomly). The algorithm then proceeds iteratively, until convergence. First, the mean and covariance parameters for the cluster representative distributions are optimally computed given the cluster assignments. These parameters are updated as shown in (8) and (9). Next, the cluster assignments $\pi$ are updated for each input Gaussian. This is done by assigning the $i^{th}$ Gaussian to the cluster $j$ with representative Gaussian that is closest in differential relative entropy.

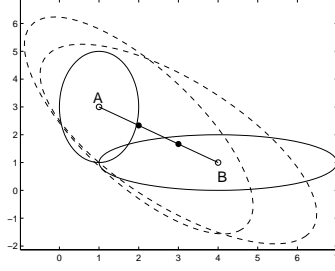

Figure 1: Optimal Gaussian representatives (shown with dotted lines) of two Gaussians centered at A and B (for two different sets of weights). While the optimal mean of each representative is the average of the individual means, the optimal covariance is the average of the individual covariances plus rank-one corrections.

Since both of these steps are locally optimal, convergence of the algorithm to a local optima can be shown. Note that the problem is $NP$-hard, so convergence to a global optima cannot be guaranteed.

We next consider the running time of Algorithm 1 when the input Gaussians are $d$-dimensional. Lines 6 and 9 compute the optimal means and covariances for each cluster, which requires $O(nd)$ and $O(nd^2)$ total work, respectively. Line 12 computes the differential relative entropy between each input Gaussian and each cluster representative Gaussian. As only the $\arg\min$ over all $\Sigma_j$ is needed, we can reduce the Burg matrix divergence computation (equation (1)) to $tr(S_i\Sigma_j^{-1}) - \log|\Sigma_j^{-1}|$. Once the inverse of each cluster covariance is computed (for a cost of $O(kd^3)$), the first term can be computed in $O(d^2)$ time. The second term can similarly be computed once for each cluster for a total cost of $O(kd^3)$. Computing the Mahalanobis distance is an $O(d^2)$ operation. Thus, total cost of line 12 is $O(kd^3 + nkd^2)$ and the total running time of the algorithm, given $\tau$ iterations, is $O(\tau kd^2(n+d))$.

---

**Algorithm 1** Differential Entropic Clustering of Multivariate Gaussians

---

1: $\{m_1, ..., m_n\} \leftarrow$ means of input Gaussians
2: $\{S_1, ..., S_n\} \leftarrow$ covariance matrices of input Gaussians
3: $\pi \leftarrow$ initial cluster assignments
4: **while** not converged **do**
5:     **for** $j = 1$ to $k$ **do** {update cluster means}
6:         $\mu_j \leftarrow \frac{1}{|\{i:\pi_i=j\}|} \sum_{i:\pi_i=j} m_i$
7:     **end for**
8:     **for** $j = 1$ to $k$ **do** {update cluster covariances}
9:         $\Sigma_j \leftarrow \frac{1}{|\{i:\pi_i=j\}|} \sum_{i:\pi_i=j} \left(S_i + (m_i - \mu_j)(m_i - \mu_j)^T\right)$
10:    **end for**
11:    **for** $i = 1$ to $n$ **do** {assign each Gaussian to the closest cluster representative Gaussian}
12:       $\pi_i \leftarrow \text{argmin}_{1 \le j \le k} B(S_i, \Sigma_j) + M_{\Sigma_j^{-1}}(m_i, \mu_j)$ {$B$ is the Burg matrix divergence and $M_{\Sigma_j^{-1}}$ is the Mahalanobis distance parameterized by $\Sigma_j$}
13:    **end for**
14: **end while**

---

## 4 Experiments

We now present experimental results for our algorithm across three different domains: a synthetic dataset, sensor network data, and a statistical debugging application.

### 4.1 Synthetic Data

Our synthetic datasets consist of a set of 200 objects, each of which consists of 30 samples drawn from one of $k$ randomly generated $d$-dimensional multivariate Gaussians. The $k$ Gaussians are

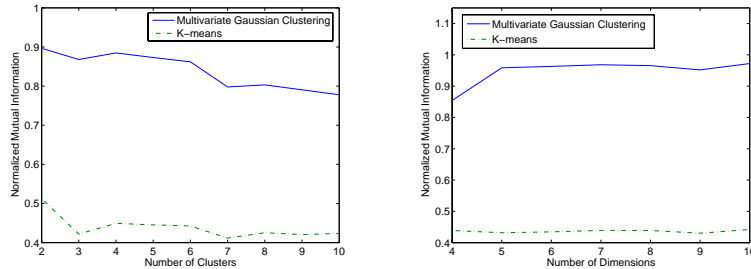

Figure 2: Clustering quality of synthetic data. Traditional $k$-means clustering uses only first-order information (i.e. the mean), whereas our Gaussian clustering algorithm also incorporates second-order covariance information. Here, we see that our algorithm achieves higher clustering quality for datasets composed of four-dimensional Gaussians with varied number of clusters (left), as well as for varied dimensionality of the input Guassians with $k = 5$ (right).

generated by choosing a mean vector uniformly at random from the unit simplex and randomly selecting a covariance matrix from the set of matrices with eigenvalues $1, 2, ..., d$.

In Figure 2, we compare our algorithm to the $k$-means algorithm, which clusters each object solely on the mean of the samples. Accuracy is quantified in terms of normalized mutual information (NMI) between discovered clusters and the true clusters, a standard technique for determining the quality of clusters. Figure 2 (left) shows the clustering quality as a function of the number of clusters when the dimensionality of the input Gaussians is fixed ($d = 4$). Figure 2 (right) gives clustering quality for five clusters across a varying number of dimensions. All results represent averaged NMI values across 50 experiments. As can be seen in Figure 2, our multivariate Gaussian clustering algorithm yields significantly higher NMI values than $k$-means for all experiments.

## 4.2 Sensor Networks

Sensor networks are wireless networks composed of small, low-cost sensors that monitor their surrounding environment. An open question in sensor networks research is how to minimize communication costs between the sensors and the base station: wireless communication requires a relatively large amount of power, a limited resource on current sensor devices (which are usually battery powered).

A recently proposed sensor network system, BBQ [4], reduces communication costs by modelling sensor network data at each sensor device using a time-varying multivariate Gaussian and transmitting only model parameters to the base station. We apply our multivariate Gaussian clustering algorithm to cluster sensor devices from the Intel Lab at Berkeley [8]. Clustering has been used in sensor network applications to determine efficient routing schemes, as well as for discovering trends between groups of sensor devices. The Intel sensor network consists of 52 working sensors, each of which monitors ambient temperature, humidity, light levels, and voltage every thirty seconds. Conditioned on time, the sensor readings can be fit quite well by a multivariate Gaussian.

Figure 3 shows the results of our multivariate Gaussian clustering algorithm applied to this sensor network data. For each device, we compute the sample mean and covariance from sensor readings between noon and 2pm each day, for 36 total days. To account for varying scales of measurement, we normalize all variables to have unit variance. The second cluster (denoted by '2' in figure 3) has the largest variance among all clusters: many of the sensors for this cluster are located in high traffic areas, including the large conference room at the top of the lab, and the smaller tables in the bottom of the lab. Since the measurements were taken during lunchtime, we expect higher traffic in these areas. Interestingly, this cluster shows very high co-variation between humidity and voltage. Cluster one is characterized by high temperatures, which is not surprising, as there are several windows on the left side of the lab. This window faces west and has an unobstructed view of the ocean. Finally, cluster three has a moderate level of total variation, with relatively low light levels. The cluster is primarily located in the center and the right of lab, away from outside windows.

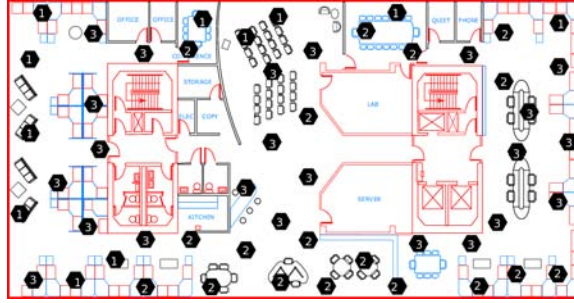

Figure 3: To reduce communication costs in sensor networks, each sensor device may be modelled by a multivariate Gaussian. The above plot shows the results of applying our algorithm to cluster sensors into three groups, denoted by labels '1', '2', and '3'.

## 4.3   Statistical Debugging

Leveraging program runtime statistics for the purpose of software debugging has received recent research attention [12]. Here we apply our algorithm to cluster functional behavior patterns over software bugs in the LaTeX document preparation program. The data is taken from the Navel system [7], a system that uses machine learning to provide better error messaging. The dataset contains four software bugs, each of which is caused by an unsuccessful LaTeX compilation (e.g. specifying an incorrect number of columns in an array environment) with ambiguous or unclear error messages provided. LaTeX has notoriously cryptic error messages for document compilation failures—for example, the message "LaTeX Error: There's no line here to end" can be caused by numerous problems in the source document.

Each function in the program's source is measured by the frequency with which it is called across each of the four software bugs. We model this distribution as a 4-dimensional multivariate Gaussian, one dimension for each bug. The distributions are estimated from a set of samples; each sample corresponds to a single LaTeX file drawn from a set of grant proposals and submitted computer science research papers. For each file and for each of the four bugs, the LaTeX compiler is executed over a slightly modified version of the file that has been changed to exhibit the bug. During program execution, function counts are measured and recorded. More details can be found in [7].

Clustering these function counts can yield important debugging insight to assist a software engineer in understanding error dependent program behavior. Figure 4 shows three covariance matrices from a sample clustering of eight clusters. To capture the dependencies between bugs, we normalize each input Gaussian to have zero mean and unit variance. Cluster (a) represents functions that are highly error independent—i.e. the matrix shows high levels of covariation among all pairs of error classes. Conversely, clusters (b) and (c) show that some functions are highly error dependent. Cluster (b) shows a high dependency between bugs 1 and 4, while cluster (c) exhibits high covariation between bugs 1 and 3, and between bugs 2 and 4.

$$
\begin{bmatrix}
1.00 & 0.94 & 0.94 & 0.94 \\
0.94 & 1.00 & 0.94 & 0.94 \\
0.94 & 0.94 & 1.00 & 0.94 \\
0.94 & 0.94 & 0.94 & 1.00
\end{bmatrix}
\quad
\begin{bmatrix}
1.00 & 0.58 & 0.58 & 0.91 \\
0.58 & 1.00 & 0.55 & 0.67 \\
0.58 & 0.55 & 1.00 & 0.68 \\
0.91 & 0.67 & 0.68 & 1.00
\end{bmatrix}
\quad
\begin{bmatrix}
1.00 & 0.58 & 0.95 & 0.58 \\
0.58 & 1.00 & 0.58 & 0.95 \\
0.95 & 0.58 & 1.00 & 0.58 \\
0.58 & 0.95 & 0.58 & 1.00
\end{bmatrix}
$$
$$(a) \qquad\qquad\qquad\qquad (b) \qquad\qquad\qquad\qquad (c)$$

Figure 4: Covariance matrices for three clusters discovered by clustering functional behavior of the LaTeX document preparation program. Cluster (a) corresponds to functions which are error-independent, while clusters (b) and (c) represent two groups of functions that exhibit different types of error dependent behavior.

## 5   Related Work

In this work, we showed that the differential relative entropy between two multivariate Gaussian distributions can be expressed as a convex combination of the Mahalanobis distance between their

mean vectors and the Burg matrix divergence between their covariances. This is in contrast to information theoretic clustering [5], where each input is taken to be a probability distribution over some finite set. In [5], no parametric form is assumed, and the Kullback-Liebler divergence (i.e. discrete relative entropy) can be computed directly from the distributions. The differential entropy between two multivariate Gaussians wass considered in [10] in the context of solving Gaussian mixture models. Although an algebraic expression for this differential entropy was given in [10], no connection to the Burg matrix divergence was made there.

Our algorithm is based on the standard expectation-maximization style clustering algorithm [6]. Although the closed-form updates used by our algorithm are similar to those employed by a Bregman clustering algorithm [1], we note that the computation of the optimal covariance matrix (equation (9)) involves the optimal mean vector.

In [9], the problem of clustering Gaussians with respect to the symmetric differential relative entropy, $D(f||g) + D(g||f)$ is considered in the context of learning HMM parameters for speech recognition. The resulting algorithm, however, is much more computationally expensive than ours; whereas in our method, the optimal means and covariance parameters can be computed via a simple closed form solution. In [9], no such solution is presented and an iterative method must instead be employed. The problem of finding the optimal Gaussian with respect to the first argument (note that equation (6) is minimized with respect to the second argument) is considered in [11] for the problem of speaker interpolation. Here, only one source is assumed, and thus clustering is not needed.

## 6 Conclusions

We have presented a new algorithm for the problem of clustering multivariate Gaussian distributions. Our algorithm is derived in an information theoretic context, which leads to interesting connections with the differential entropy between multivariate Gaussians, and Bregman divergences. Unlike existing clustering algorithms, our algorithm optimizes both first and second order information in the data. We have demonstrated the use of our method on sensor network data and a statistical debugging application.

## References

[1] A. Banerjee, S. Merugu, I. Dhillon, and S. Ghosh. Clustering with Bregman divergences. In *Siam International Conference on Data Mining*, pages 234–245, 2004.

[2] L. Bregman. The relaxation method finding the common point of convex sets and its application to the solutions of problems in convex programming. In *USSR Comp. of Mathematics and Mathematical Physics*, volume 7, pages 200–217, 1967.

[3] T. M. Cover and J. A. Thomas. *Elements of information theory*. Wiley Series in Telecommunications, 1991.

[4] A. Deshpande, C. Guestrin, S. Madden, J. Hellerstein, and W. Hong. Model-based approximate querying in sensor networks. In *International Journal of Very Large Data Bases*, 2005.

[5] I. Dhillon, S. Mallela, and R. Kumar. A divisive information-theoretic feature clustering algorithm for text classification. In *Journal of Machine Learning Research*, volume 3, pages 1265–1287, 2003.

[6] R. O. Duda, P. E. Hart, and D. G. Stork. *Pattern Classification*. John Wiley and Sons, Inc., 2001.

[7] J. Ha, H. Ramadan, J. Davis, C. Rossbach, I. Roy, and E. Witchel. Navel: Automating software support by classifying program behavior. Technical Report TR-06-11, University of Texas at Austin, 2006.

[8] S. Madden. Intel lab data. http://berkeley.intel-research.net/labdata, 2004.

[9] T. Myrvoll and F. Soong. On divergence based clustering of normal distributions and its application to HMM adaptation. In *Eurospeech*, pages 1517–1520, 2003.

[10] Y. Singer and M. Warmuth. Batch and on-line parameter estimation of Gaussian mixtures based on the joint entropy. In *Neural Information Processing Systems*, 1998.

[11] T. Yoshimura, T. Masuko, K. Tokuda, T. Kobayashi, and T. Kitamura. Speaker interpolation in HMM-based speech synthesis. In *European Conference on Speech Communication and Technology*, 1997.

[12] A. Zheng, M. Jordan, B. Liblit, and A. Aiken. Statistical debugging of sampled programs. In *Neural Information Processing Systems*, 2004.
